# Processing of Time Series by Neural Circuits with Biologically Realistic Synaptic Dynamics

**Thomas Natschläger & Wolfgang Maass**
Institute for Theoretical Computer Science
Technische Universität Graz, Austria
{tnatschl,maass}@igi.tu-graz.ac.at

**Eduardo D. Sontag**
Dept. of Mathematics
Rutgers University
New Brunswick, NJ 08903, USA
sontag@hilbert.rutgers.edu

**Anthony Zador**
Cold Spring Harbor Laboratory
1 Bungtown Rd
Cold Spring Harbor, NY 11724
zador@cshl.org

## Abstract

Experimental data show that biological synapses behave quite differently from the symbolic synapses in common artificial neural network models. Biological synapses are dynamic, i.e., their "weight" changes on a short time scale by several hundred percent in dependence of the past input to the synapse. In this article we explore the consequences that these synaptic dynamics entail for the computational power of feedforward neural networks. We show that gradient descent suffices to approximate a given (quadratic) filter by a rather small neural system with dynamic synapses. We also compare our network model to artificial neural networks designed for time series processing. Our numerical results are complemented by theoretical analysis which show that even with just a single hidden layer such networks can approximate a surprisingly large large class of nonlinear filters: all filters that can be characterized by Volterra series. This result is robust with regard to various changes in the model for synaptic dynamics.

## 1 Introduction

More than two decades of research on artificial neural networks has emphasized the central role of synapses in neural computation. In a conventional artificial neural network, all units ("neurons") are assumed to be identical, so that the computation is completely specified by the synaptic "weights," *i.e.* by the strengths of the connections between the units. Synapses in common artificial neural network models are static: the value $w_{ij}$ of a synaptic weight is assumed to change only during "learning". In contrast to that, the "weight" $w_{ij}(t)$ of a biological synapse at time $t$ is known to be strongly dependent on the inputs $x_j(t - \tau)$ that this synapse has received from the presynaptic neuron $i$ at previous time steps $t - \tau$, see e.g. [1]. We will focus in this article on mean-field models for populations of neurons connected by dynamic synapses.

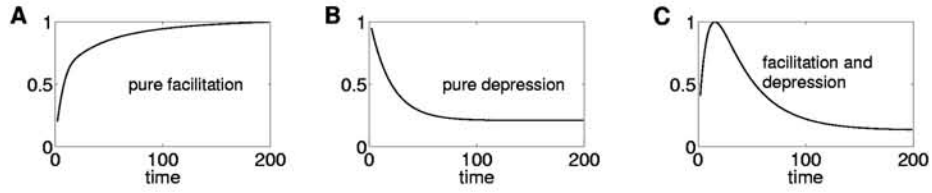

Figure 1: A dynamic synapse can produce quite different outputs for the same input. The response of a single synapse to a step increase in input activity applied at time step 0 is compared for three different parameter settings.

Several models for single synapses have been proposed for the dynamic changes in synaptic efficacy. In [2] the model of [3] is extended to populations of neurons where the current synaptic efficacy $w_{ij}(t)$ between a population $j$ and a population $i$ at time $t$ is modeled as a product of a facilitation term $f_{ij}(t)$ and a depression term $d_{ij}(t)$ scaled by the factor $W_{ij}$. We consider a time discrete version of this model defined as follows:

$$w_{ij}(t) = W_{ij} \cdot f_{ij}(t) \cdot d_{ij}(t) \tag{1}$$

$$\bar{f}_{ij}(t+1) = \bar{f}_{ij}(t) - \frac{\bar{f}_{ij}(t)}{F_{ij}} + U_{ij} \cdot (1 - \bar{f}_{ij}(t)) \cdot x_j(t) \tag{2}$$

$$d_{ij}(t+1) = d_{ij}(t) + \frac{1 - d_{ij}(t)}{D_{ij}} - f_{ij}(t) \cdot d_{ij}(t) \cdot x_j(t) \tag{3}$$

$$f_{ij}(t) = \bar{f}_{ij}(t) \cdot (1 - U_{ij}) + U_{ij} \tag{4}$$

with $d_{ij}(0) = 1$ and $f_{ij}(0) = 0$. Equation (2) models facilitation (with time constant $F_{ij}$), whereas equation (3) models the combined effects of synaptic depression (with time constant $D_{ij}$) and facilitation. Depending on the values of the characteristic parameters $U_{ij}$, $D_{ij}$, $F_{ij}$ a synaptic connection $\langle ij \rangle$ maps an input function $x_j(t)$ into the corresponding time varying synaptic output $w_{ij}(t) \cdot x_j(t)$. The same input $x_j(t)$ can yield markedly different outputs $w_{ij}(t) \cdot x_i(t)$ for different values of the characteristic parameters $U_{ij}$, $D_{ij}$, $F_{ij}$. Fig. 1 compares the output for three different sets of values for the parameters $U_{ij}$, $D_{ij}$, $F_{ij}$. These examples illustrate just three of the range of input-output behaviors that a single synapse can achieve.

In this article we will consider feedforward networks coupled by dynamic synapses. One should think of the computational units in such a network as populations of spiking neurons. We refer to such networks as *"dynamic networks"*, see Fig. 2 for details.

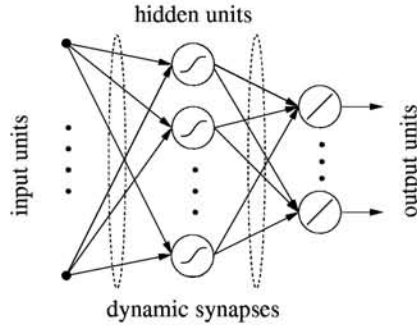

hidden units

input units

output units

dynamic synapses

Figure 2: The dynamic network model. The output $x_i(t)$ of the $i^{th}$ unit is given by $x_i(t) = \sigma(\sum_j w_{ij}(t) \cdot x_j(t))$, where $\sigma$ is either the sigmoid function $\sigma(u) = 1/(1 + \exp(-u))$ (in the hidden layers) or just the identity function $\sigma(u) = u$ (in the output layer) and $w_{ij}(t)$ is modeled according to Equ. (1) to (4).

In Sections 2 and 3 we demonstrate (by employing gradient descent to find appropriate values for the parameters $U_{ij}$, $D_{ij}$, $F_{ij}$ and $W_{ij}$) that even small dynamic networks can compute complex quadratic filters. In Section 4 we address the question which synaptic parameters are important for a dynamic network to learn a given filter. In Section 5 we give a precise mathematical characterization of the computational power of such dynamic networks.

## 2  Learning Arbitrary Quadratic Filters by Dynamic Networks

In order to analyze which filters can be approximated by small dynamic networks we investigate the task of learning a quadratic filter $Q$ randomly chosen from a class $\mathcal{Q}_m$. The class $\mathcal{Q}_m$ consists of all quadratic filters $Q$ whose output $(Qx)(t)$ in response to the input time series $x(t)$ is defined by some symmetric $m \times m$ matrix $H_Q = [h_{kl}]$ of filter coefficients $h_{kl} \in \mathbb{R}$, $k = 1 \ldots m$, $l = 1 \ldots m$ through the equation $(Qx)(t) = \sum_{l=1}^{m} \sum_{k=1}^{m} h_{kl} \; x(t-k) \, x(t-l)$ . An example of the input and output for one choice of quadratic parameters ($m = 10$) are shown in Figs. 3B and 3C, respectively. We view such filter $Q$ as an example for the kinds of complex transformations that are important to an organism's survival, such as those required for motor control and the processing of time-varying sensory inputs. For example, the spectrotemporal receptive field of a neuron in the auditory cortex [4] reflects some complex transformation of sound pressure to neuronal activity. The real transformations actually required may be very complex, but the simple filter $Q$ provides a useful starting point for assessing the capacity of this architecture to transform one time-varying signal into another.

Can a network of units coupled by dynamic synapses implement the filter $Q$? We tested the approximation capabilities of a rather small dynamic network with just 10 hidden units (5 excitatory and 5 inhibitory ones), and one output (Fig. 3A). The dynamics of inhibitory synapses is described by the same model as that for excitatory synapses. For any particular temporal pattern applied at the input and any particular choice of the synaptic parameters, this network generates a temporal pattern as output. This output can be thought of, for example, as the activity of a particular population of neurons in the cortex, and the target function as the time series generated for the same input by some unknown quadratic filter $Q$. The synaptic parameters $W_{ij}$, $D_{ij}$, $F_{ij}$ and $U_{ij}$ are chosen so that, for each input in the training set, the network minimized the mean-square error $E[z, z_Q] = \frac{1}{T} \sum_{t=0}^{T-1} (z(t) - z_Q(t))^2$ between its output $z(t)$ and the desired output $z_Q(t)$ specified by the filter $Q$. To achieve this minimization, we used a conjugate gradient algorithm.[1] The training inputs were random signals, an example of which is shown in Fig. 3B. The test inputs were drawn from the same random distribution as the training inputs, but were not actually used during training. This test of generalization ensured that the observed performance represented more than simple "memorization" of the training set. Fig. 3C compares the network performance before and after training. Prior to training, the output is nearly flat, while after training the network output tracks the filter output closely ($E[z, z_Q] = 0.0032$).

Fig. 3D shows the performance after training for different randomly chosen quadratic filters $Q \in \mathcal{Q}_m$ for $m = 4, \ldots, 16$. Even for larger values of $m$ the relatively small network with 10 hidden units performs rather well. Note that a quadratic filter of dimension $m$ has $m(m + 1)/2$ free parameters, whereas the dynamic network has a constant number of 80 adjustable parameters. This shows clearly that dynamic synapses enable a small network to mimic a wide range of possible quadratic target filters.

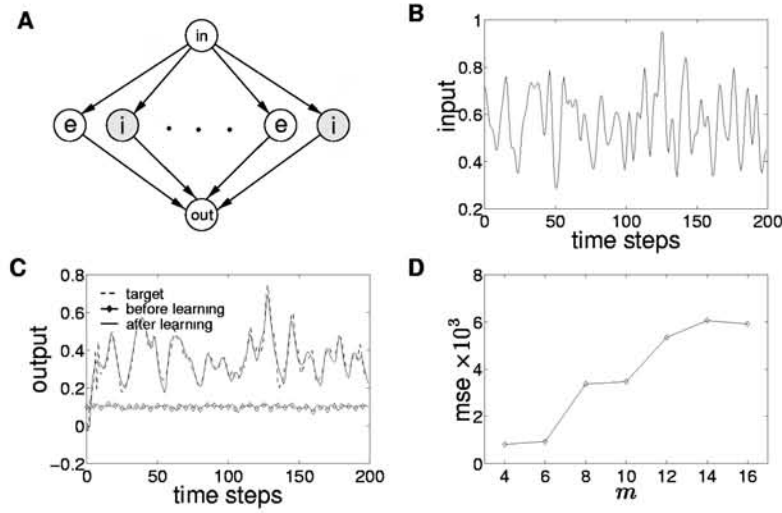

Figure 3: A network with units coupled by dynamic synapses can approximate randomly drawn quadratic filters. **A** Network architecture. The network had one input unit, 10 hidden units (5 excitatory, 5 inhibitory), and one output unit, see Fig. 2 for details. **B** One of the input patterns used in the training ensemble. For clarity, only a portion of the actual input is shown. **C** Output of the network prior to training, with random initialization of the parameters, and the output of the dynamic network after learning. The target was the output of a quadratic filter $Q \in \mathcal{Q}_{10}$. The filter coefficients $h_{kl}$ ($1 \leq k, l \leq 10$) were generated randomly by subtracting $\mu/2$ from a random number generated from an exponential distribution with mean $\mu = 3$. **D** Performance after network training. For different sizes of $H_Q$ ($H_Q$ is a symmetric $m \times m$ matrix) we plotted the average performance (mse measured on a test set) over 20 different filters $Q$, i.e. 20 randomly generated matrices $H_Q$.

## 3   Comparison with the model of Back and Tsoi

Our dynamic network model is not the first to incorporate temporal dynamics via dynamic synapses. Perhaps the earliest suggestion for a role for synaptic dynamics in network computation was by [7]. More recently, a number of networks have been proposed in which synapses implemented linear filters; in particular [6].

To assess the performance of our network model in relation to the model proposed in [6] we have analyzed the performance of our dynamic network model for the same system identification task that was employed as benchmark task in [6]. The goal of this task is to learn a filter $F$ with $(Fx)(t) = \sin(u(t))$ where $u(t)$ is the output of a linear filter applied to the input time series $x(t)$.[2]

The result is summarized in Fig. 4. It can clearly be seen that our network model (see Fig. 3A for the network architecture) is able to learn this particular filter. The mean square error (mse) on the test data is 0.0010, which is slightly smaller than the mse of 0.0013 reported in [6]. Note that the network Back and Tsoi used to learn the task had 130 adjustable parameters (13 parameters per IIR synapse, 10 hidden units) whereas our network model had only 80 adjustable parameters (all parameters $U_{ij}$, $F_{ij}$, $D_{ij}$ and $W_{ij}$ were adjusted during learning).

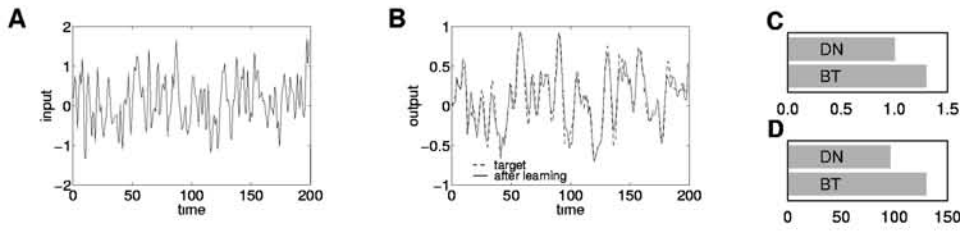

Figure 4: Performance of our model on the system identification task used in [6]. The network architecture is the same as in Fig. 3. **A** One of the input patterns used in the training ensemble. **B** Output of the network after learning and the target. **C** Comparison of the mean square error (in units of $10^{-3}$) achieved on test data by the model of Back and Tsoi (BT) and by the dynamic network (DN). **D** Comparison of the number of adjustable parameters. The network model of Back and Tsoi (BT) utilizes slightly more adjustable parameters than the dynamic network (DN).

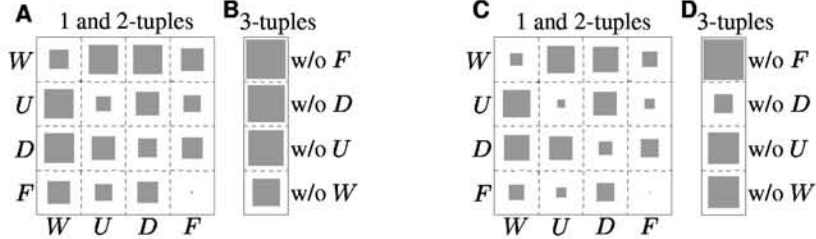

Figure 5: Impact of different synaptic parameters on the learning capabilities of a dynamic network. The size of a square (the "impact") is proportional to the inverse of the mean squared error averaged over $N$ trials. **A** In each trial ($N = 100$) a different quadratic filter matrix $H_Q$ ($m = 6$) was randomly generated as described in Fig. 3. Along the diagonal one can see the impact of a single parameter, whereas the off-diagonal elements (which are symmetric) represent the impact of changing pairs of parameters. **B** The impact of subsets of size three is shown where the labels indicate which parameter is not included. **C** Same interpretation as for panel A but the results shown ($N = 20$) are for the filter used in [6]. **D** Same interpretation as for panel B but the results shown ($N = 20$) are for the same filter as in panel C.

This shows that a very simple feedforward network with biologically realistic synaptic dynamics yields performance comparable to that of artificial networks that were previously designed to yield good performance in the time series domain without any claims of biological realism.

## 4 Which Parameters Matter?

It remains an open experimental question which synaptic parameters are subject to use-dependent plasticity, and under what conditions. For example, long term potentiation appears to change synaptic dynamics between pairs of layer 5 cortical neurons [8] but not in the hippocampus [9]. We therefore wondered whether plasticity in the synaptic dynamics is essential for a dynamic network to be able to learn a particular target filter. To address this question, we compared network performance when different parameter subsets were optimized using the conjugate gradient algorithm, while the other parameters were held fixed. In all experiments, the fixed parameters were chosen to ensure heterogeneity in presynaptic dynamics.

Fig. 5 shows that changing only the postsynaptic parameter $W$ has comparable impact to changing only the presynaptic parameters $U$ or $D$, whereas changing only $F$ has little impact on the dynamics of these networks (see diagonal of Fig. 5A and Fig. 5C). However, to achieve good performance one has to change at least two different types of parameters such as $\{W, U\}$ or $\{W, D\}$ (all other pairs yield worse performance). Hence, neither plasticity in the presynaptic dynamics $(U, D, F)$ alone nor plasticity of the postsynaptic efficacy $(W)$ alone was sufficient to achieve good performance in this model.

## 5  A Universal Approximation Theorem for Dynamic Networks

In the preceding sections we had presented empirical evidence for the approximation capabilities of our dynamic network model for computations in the time series domain. This gives rise to the question, what the theoretical limits of their approximation capabilities are. The rigorous theoretical result presented in this section shows that basically there are no significant a priori limits. Furthermore, in spite of the rather complicated system of equations that defines dynamic networks, one can give a precise mathematical characterization of the class of filters that can be approximated by them. This characterization involves the following basic concepts. An arbitrary filter $F$ is called *time invariant* if a shift of the input functions by a constant $t_0$ just causes a shift of the output function by the same constant $t_0$. Another essential property of filters is *fading memory*. A filter $F$ has fading memory if and only if the value of $F\underline{x}(0)$ can be approximated arbitrarily closely by the value of $F\tilde{\underline{x}}(0)$ for functions $\tilde{\underline{x}}$ that approximate the functions $\underline{x}$ for sufficiently long bounded intervals $[-T, 0]$. Interesting examples of linear and nonlinear time invariant filters with fading memory can be generated with the help of representations of the form $(Fx)(t) = \int_0^\infty \ldots \int_0^\infty x(t - \tau_1) \cdot \ldots \cdot x(t - \tau_k) h(\tau_1, \ldots, \tau_k) d\tau_1 \ldots d\tau_k$ for measurable and essentially bounded functions $x : \mathbb{R} \to \mathbb{R}$ (with $h \in L^1$). One refers to such an integral as a *Volterra term of order $k$*. Note that for $k = 1$ it yields the usual representation for a *linear* time invariant filter. The class of filters that can be represented by Volterra series, i.e., by finite or infinite sums of Volterra terms of arbitrary order, has been investigated for quite some time in neurobiology and engineering.

**Theorem 1** *Assume that $X$ is the class of functions from $\mathbb{R}$ into $[B_0, B_1]$ which satisfy $|x(t) - x(s)| \leq B_2 \cdot |t - s|$ for all $t, s \in \mathbb{R}$, where $B_0, B_1, B_2$ are arbitrary real-valued constants with $0 < B_0 < B_1$ and $0 < B_2$. Let $F$ be an arbitrary filter that maps vectors of functions $\underline{x} = \langle x_1, \ldots, x_n \rangle \in X^n$ into functions from $\mathbb{R}$ into $\mathbb{R}$. Then the following are equivalent:*

   *(a) $F$ can be approximated by dynamic networks $\mathcal{N}$ defined in Fig. 2 (i.e., for any $\varepsilon > 0$ there exists such network $\mathcal{N}$ such that $|(F\underline{x})(t) - (\mathcal{N}\underline{x})(t)| < \varepsilon$ for all $\underline{x} \in X^n$ and all $t \in \mathbb{R}$)*

   *(b) $F$ can be approximated by dynamic networks (see Fig. 2) with just a single layer of sigmoidal neurons*

   *(c) $F$ is time invariant and has fading memory*

   *(d) $F$ can be approximated by a sequence of (finite or infinite) Volterra series.*

The *proof* of Theorem 1 relies on the Stone-Weierstrass Theorem, and is contained as the proof of Theorem 3.4 in [10].

The *universal approximation result* contained in Theorem 1 turns out to be rather robust with regard to changes in the definition of a dynamic network. Dynamic networks with just one layer of dynamic synapses and one subsequent layer of sigmoidal gates can approximate the same class of filters as dynamic networks with an arbitrary number of layers of

dynamic synapses and sigmoidal neurons. It can also be shown that Theorem 1 remains valid if one considers networks which have depressing synapses only or if one uses the model for synaptic dynamics proposed in [1].

## 6  Discussion

Our central hypothesis is that rapid changes in synaptic strength, mediated by mechanisms such as facilitation and depression, are an integral part of neural processing. We have analyzed the computational power of such *dynamic networks*, which represent a new paradigm for neural computation on time series that is based on biologically realistic models for synaptic dynamics [11].

Our analytical results show that the class of nonlinear filters that can be approximated by dynamic networks, even with just a single hidden layer of sigmoidal neurons, is remarkably rich. It contains every time invariant filter with fading memory, hence arguable every filter that is potentially useful for a biological organism.

The computer simulations we performed show that rather small dynamic networks are not only able to perform interesting computations on time series, but their performance is comparable to that of previously considered artificial neural networks that were designed for the purpose of yielding efficient processing of temporal signals. We have tested dynamic networks on tasks such as the learning of a randomly chosen quadratic filter, as well as on the learning task used in [6], to illustrate the potential of this architecture.

## Footnotes

[1] In order to apply such a conjugate gradient algorithm ones has to calculate the partial derivatives $\frac{\delta E[z, z_Q]}{\delta U_{ij}}$, $\frac{\delta E[z, z_Q]}{\delta D_{ij}}$, $\frac{\delta E[z, z_Q]}{\delta F_{ij}}$ and $\frac{\delta E[z, z_Q]}{\delta W_{ij}}$ for all synapses $\langle ij \rangle$ in the network. For more details about conjugate gradient algorithms see e.g. [5].

[2]$u(t)$ is the solution to the difference equation $u(t)-1.99u(t-1)+1.572u(t-2\,1)-0.4583u(t-3\,1) = 0.0154x(t) + 0.0462x(t-1) + 0.0462x(t-2\,1) + 0.0154x(t-3\,1)$. Hence, $u(t)$ is the output of a linear filter applied to the input $x(t)$.

## References

[1] J. A. Varela, K. Sen, J. Gibson, J. Fost, L. F. Abbott, and S. B. Nelson. A quantitative description of short-term plasticity at excitatory synapses in layer 2/3 of rat primary visual cortex. *J. Neurosci*, 17:220–4, 1997.

[2] M.V. Tsodyks, K. Pawelzik, and H. Markram. Neural networks with dynamic synapses. *Neural Computation*, 10:821–835, 1998.

[3] H. Markram, Y. Wang, and M. Tsodyks. Differential signaling via the same axon of neocortical pyramidal neurons. *PNAS*, 95:5323–5328, 1998.

[4] R.C. deCharms and M.M. Merzenich. Optimizing sound features for cortical neurons. *Science*, 280:1439–43, 1998.

[5] John Hertz, Anders Krogh, and Richard Palmer. *Introduction to the Theory of Neural Computation*. Addison-Wesley, 1991.

[6] A. D. Back and A. C. Tsoi. A simplified gradient algorithm for IIR synapse multilayer perceptrons. *Neural Computation*, 5:456–462, 1993.

[7] W.A. Little and G.L. Shaw. A statistical theory of short and long term memory. *Behavioural Biology*, 14:115–33, 1975.

[8] H. Markram and M. Tsodyks. Redistribution of synaptic efficacy between neocortical pyramidal neurons. *Nature*, 382:807–10, 1996.

[9] D.K. Selig, R.A. Nicoll, and R.C. Malenka. Hippocampal long-term potentiation preserves the fidelity of postsynaptic responses to presynaptic bursts. *J. Neurosci.*, 19:1236–46, 1999.

[10] W. Maass and E. D. Sontag. Neural systems as nonlinear filters. *Neural Computation*, 12(8):1743–1772, 2000.

[11] A. M. Zador. The basic unit of computation. *Nature Neuroscience*, 3(Supp):1167, 2000.
